# Independent Component Analysis of Intracellular Calcium Spike Data

**Klaus Prank, Julia Börger, Alexander von zur Mühlen,**
**Georg Brabant, Christof Schöfl**
Department of Clinical Endocrinology
Medical School Hannover
D-30625 Hannover
Germany

## Abstract

Calcium ($Ca^{2+}$)is an ubiquitous intracellular messenger which regulates cellular processes, such as secretion, contraction, and cell proliferation. A number of different cell types respond to hormonal stimuli with periodic oscillations of the intracellular free calcium concentration ($[Ca^{2+}]_i$). These $Ca^{2+}$ signals are often organized in complex temporal and spatial patterns even under conditions of sustained stimulation. Here we study the spatio-temporal aspects of intracellular calcium ($[Ca^{2+}]_i$) oscillations in clonal $\beta$-cells (hamster insulin secreting cells, HIT) under pharmacological stimulation (Schöfl *et al.*, 1996). We use a novel fast fixed-point algorithm (Hyvärinen and Oja, 1997) for *Independent Component Analysis* (ICA) to blind source separation of the spatio-temporal dynamics of $[Ca^{2+}]_i$ in a HIT-cell. Using this approach we find two significant independent components out of five differently mixed input signals: one $[Ca^{2+}]_i$ signal with a mean oscillatory period of $68s$ and a high frequency signal with a broadband power spectrum with considerable spectral density. This results is in good agreement with a study on high-frequency $[Ca^{2+}]_i$ oscillations (Paluš *et al.*, 1998) Further theoretical and experimental studies have to be performed to resolve the question on the functional impact of intracellular signaling of these independent $[Ca^{2+}]_i$ signals.

# 1   INTRODUCTION

*Independent component analysis (ICA)* (Comon, 1994; Jutten and Herault, 1991)
has recently received much attention as a signal processing method which has been
successfully applied to blind source separation and feature extraction. The goal of
ICA is to find independent sources in an unknown linear mixture of measured sen-
sory data. This goal is obtained by reducing 2nd-order and higher order statistical
dependencies to make the signals as independent as possible. Mainly three different
approaches for ICA exist. The first approach is based on batch computations min-
imizing or maximizing some relevant criterion functions (Cardoso, 1992; Comon,
1994). The second category contains adaptive algorithms often based on stochastic
gradient methods, which may have implementations in neural networks (Amari *et
al.*, 1996; Bell and Sejnowski, 1995; Delfosse and Loubaton, 1995; Hyvärinen and
Oja, 1996; Jutten and Herault, 1991; Moreau and Macchi, 1993; Oja and Karhunen,
1995). The third class of algorithms is based on a *fixed-point iteration scheme* for
finding the local extrema of the kurtosis of a linear combination of the observed
variables which is equivalent to estimating the non-Gaussian independent compo-
nents (Hyvärinen and Oja 1997). Here we use the *fast fixed-point algorithm* for
independent component analysis proposed by Hyvärinen and Oja (1997) to analyze
the spatio-temporal dynamics of intracellular free calcium ($[Ca^{2+}]_i$) in a hamster
insulin secreting cell (HIT).

Oscillations of $[Ca^{2+}]_i$ have been reported in a number of electrically excitable and
non-excitable cells and the hypotheses of frequency coding were proposed a decade
ago (Berridge and Galione, 1988). Recent experimental results clearly demonstrate
that $[Ca^{2+}]_i$ oscillations and their frequency can be specific for gene activation con-
cerning the efficiency as well as the selectivity (Dolmetsch *et al.*, 1998). Cells are
highly compartmentalized structures which can not be regarded as homogenous en-
tities. Thus, $[Ca^{2+}]_i$ oscillations do not occur uniformly throughout the cell but
are initiated at specific sites which are distributed in a functional and nonunifortm
manner. These $[Ca^{2+}]_i$ oscillations spread across individual cells in the form of
$Ca^{2+}$ waves. $[Ca^{2+}]_i$ gradients within cells have been proposed to initiate cell mi-
gration, exocytosis, lymphocyte, killer cell activity, acid secretion, transcellular ion
transport, neurotransmitter release, gap junction regulation, and numerous other
functions (Tsien and Tsien, 1990). Due to this fact it is of major importance to
study the spatio-temporal aspects of $[Ca^{2+}]_i$ signaling in small subcompartments
using calcium-specific fluorescent reporter dyes and digital videomicroscopy rather
than studying the cell as a uniform entity. The aim of this study was to define the
independent components of the spatio-temporal $[Ca^{2+}]_i$ signal.

# 2   METHODS

## 2.1   FAST FIXED-POINT ALGORITHM USING KURTOSIS FOR
        INDEPENDENT COMPONENT ANALYSIS

In *Independent Component Analysis (ICA)* the original independent sources are un-
known. In this study we have recorded the $[Ca^{2+}]_i$ signal in single HIT-cells under
pharmacological stimulation at different subcellular regions ($m = 5$) in parallel.
The $[Ca^{2+}]_i$ signals (mixtures of sources) are denoted as $x_1, x_2, \ldots, x_m$. Each $x_i$
is expressed as the weighted sum of $n$ unknown statistically independent compo-

nents (ICs), denoted as $s_1, s_2, \ldots, s_n$. The components are assumed to be mutually statistically independent and zero-mean. The measured signals $x_i$ as well as the independent component variables can be arranged into vectors $\mathbf{x} = (\mathbf{x_1}, \mathbf{x_2}, \ldots, \mathbf{x_m})$ and $\mathbf{s} = (\mathbf{s_1}, \mathbf{s_2}, \ldots, \mathbf{s_n})$ respectively. The linear relationship is given by:

$$\mathbf{x} = \mathbf{As} \tag{1}$$

Here $\mathbf{A}$ is a constant mixing matrix whose elements $a_{ij}$ are the unknown coefficients of the mixtures. The basic problem of ICA is to estimate both the mixing matrix $\mathbf{A}$ and the realizations of the $s_i$ using only observations of the mixtures $x_j$. In order to perform ICA, it is necessary to have at least as many mixtures as there are independent sources ($m \geq n$). The assumption of zero mean of the ICs is no restriction, as this can always be accomplished by subtracting the mean from the random vector $\mathbf{x}$. The ICs and the columns of $\mathbf{A}$ can only be estimated up to a multiplicative constant, because any constant multiplying an IC in eq. 1 could be cancelled by dividing the corresponding column of the mixing matrix $\mathbf{A}$ by the same constant. For mathematical convenience, the ICs are defined to have unit variance making the (non-Gaussian) ICs unique, up to their signs (Comon, 1994). Here we use a novel *fixed-point algorithm* for ICA estimation which is based on 'contrast' functions whose extrema are closely connected to the estimation of ICs (Hyvärinen and Oja, 1997). This method denoted as *fast fixed-point algorithm* has a number of desirable properties. First, it is easy to use, since there are no user-defined parameters. Furthermore, the convergence is fast, conventionally in less than 15 steps and for an appropriate contrast function, the fixed-point algorithm is much more robust against outliers than most ICA algorithms.

Most solutions to the ICA problem use the fourth-order cumulant or *kurtosis* of the signals, defined for a zero-mean random variable $x$ as:

$$kurt(x) = E\{x^4\} - 3(E\{x^2\})^2, \tag{2}$$

where $E\{x\}$ denotes the mathematical expectation of $x$. The kurtosis is negative for source signals whose amplitude has sub-Gaussian probability densitites (distribution flatter than Gaussian, positive for super Gaussian) sharper than Gaussian, and zero for Gausssian densities. Kurtosis is a contrast function for ICA in the following sense. Consider a linear combination of the measured mixtures $\mathbf{x}$, say $\mathbf{w}^T\mathbf{x}$, where the vector $\mathbf{w}$ is constrained so that $E\{(w^T x)^2\} = 1$. When $w^T x = \pm s_i$, for some $i$, i.e. when the linear combination equals, up to the sign, one of the ICs, the kurtosis of $w^T x$ is locally minimized or maximized. This property is widely used in ICA algorithms and forms the basis of the *fixed-point algorithm* used in this study which finds the relevant extrema of kurtosis also for non-whitened data. Based on this fact, Hyvärinen and Oja (1997) introduced a very simple and highly efficient *fixed-point algorithm* for computing ICA, calculated over sphered zero-mean vectors $\mathbf{x}$, that is able to find the rows of the separation matrix (denoted as $\mathbf{w}$) and so identify one independent source at a time. The algorithm which computes a gradient descent over the kurtosis is defined as follows:

1. Take a random initial vector $w_0$ of unit norm. Let $l = 1$.

2. Let $w_l = E\{v(w_{l-1}^T v)^3\} - 3w_{l-1}$. The expectation can be estimated using a large sample of $v_k$ vectors.

3. Divide $w_l$ by its norm (e.g. the Euclidean norm $\| w \| = \sqrt{\sum_i w_i^2}$).

4. If $| w_l^T w_{l-1} |$ is not close enough to 1, let $l = l + 1$ and go back to step 2. Otherwise, output the vector $w_l$.

To calculate more than one solution, the algorithm may be run as many times as required. It is nevertheless, necessary to remove the information contained in the solutions already found, to estimate each time a different independent component. This can be achieved, after the fourth step of the algorithm, by simply subtracting the estimated solution $\hat{s} = w^T v$ from the unsphered data **x**.

In the first step of analysis we determined the eigenvalues of the covariance matrix of the measured $[Ca^{2+}]_i$ signals $s_i$ to reduce the dimensionality of the system. Then the *fast fixed-point algorithm* was run using the experimental $[Ca^{2+}]_i$ data to determine the ICs. The resulting ICs were analyzed in respect to their frequency content by computing the Fourier power spectrum.

## 2.2 MEASUREMENT OF INTRACELLULAR CALCIUM IN HIT-CELLS

To measure $[Ca^{2+}]_i$, HIT (hamster insulin secreting tumor)-cells were loaded with the fluorescent indicator Fura-2/AM and Fura-2 fluorescence was recorded at five different subcellular regions in parallel using a dual excitation spectrofluorometer videoimaging system. The emission wavelength was 510 nm and the excitation wavelengths were 340 nm and 380 nm respectively. The ration between the excitation wavelength ($F_{340nm}/F_{380nm}$) which correlates to $[Ca^{2+}]_i$ was sampled at a rate of 1 Hz over 360 s. $[Ca^{2+}]_i$ spikes in this cell were induced by the administration of 1 nM arginine vasopressin (AVP).

## 3 RESULTS

From the five experimental $[Ca^{2+}]_i$ signals (Fig. 1) we determined two significant eigenvalues of the covariance matrix. The *fixed-point algorithm* converged in less than 15 steps and yielded two different ICs, one slowly oscillating component with a mean period of 68 s and one component with fast irregular oscillations with a flat broadband power spectrum (Fig. 2). The spectral density of the second component was considerably larger than that for the high-frequency content of the first slowly oscillating component.

## 4 CONCLUSIONS

Changes in $[Ca^{2+}]_i$ associated with $Ca^{2+}$ oscillations generally do not occur uniformly throughout the cell but are initiated at specific sites and are able to spread across individual cells in the form of intracellular $Ca^{2+}$ waves. Furthermore, $Ca^{2+}$ signaling is not limited to single cells but occurs between adjacent cells in the form of intercellular $Ca^{2+}$ waves. The reasons for these spatio-temporal patterns of $[Ca^{2+}]_i$ are not yet fully understood. It has been suggested that information is encoded in the frequency, rather than the amplitude, of $Ca^{2+}$ oscillations, which has the advantage of avoiding prolonged exposures to high $[Ca^{2+}]_i$. Another advantage of

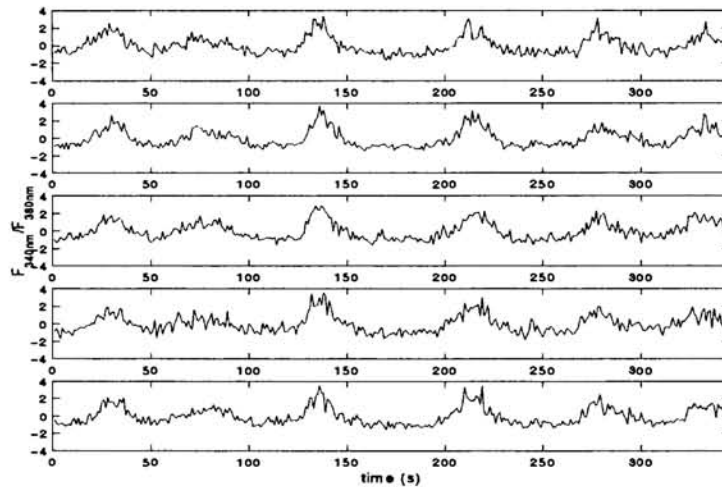

Figure 1: Experimental time series of $[Ca^{2+}]_i$ in a $\beta$-cell (insulin secreting cell from a hamster, HIT-cell) determined in five subcellular regions. The data are given as the ratio between both exciation wavelengths of 340 nm and 380 nm respectively which correspond to $[Ca^{2+}]_i$. $[Ca^{2+}]_i$ can be calculated from this ratio. The plotted time series are whitened.

frequency modulated signaling is its high signal-to-noise ratio. In the spatial domain, the spreading of a $Ca^{2+}$ oscillation as a $Ca^{2+}$ wave provides a mechanism by which the regulatory signal can be distributed throughout the cell. The extension of $Ca^{2+}$ waves to adjacent cells by intercellular communication provides one mechanism by which multicellular systems can effect coordinated and cooperative cell responses to localized stimuli. In this study we demonstrated that the $[Ca^{2+}]_i$ signal in clonal $\beta$−cells (HIT cells) is composed of two independent components using spatio-temporal $[Ca^{2+}]_i$ data for analysis. One component can be described as large amplitude slow frequency oscillations whereas the other one is a high frequency component which exhibits a broadband power spectrum. These results are in good agreement with a previous study where only the temporal dynamics of $[Ca^{2+}]_i$ in HIT cells has been studied. Using coarse-grained entropy rates computed from information-theoretic functionals we could demonstrate in that study that a fast oscillatory component of the $[Ca^{2+}]_i$ signal can be modulated pharmacologically suggesting deterministic structure in the temporal dynamics (Paluš et al., 1998). Since $Ca^{2+}$ is central to the stimulation of insulin secretion from pancreatic $\beta$-cells future experimental and theoretical studies should evaluate the impact of the different oscillatory components of $[Ca^{2+}]_i$ onto the secretory process as well as gene transcription. One possibility to resolve that question is to use a recently proposed mathematical model which allows for the on-line decoding of the $[Ca^{2+}]_i$ into the cellular response represented by the activation (phospho-

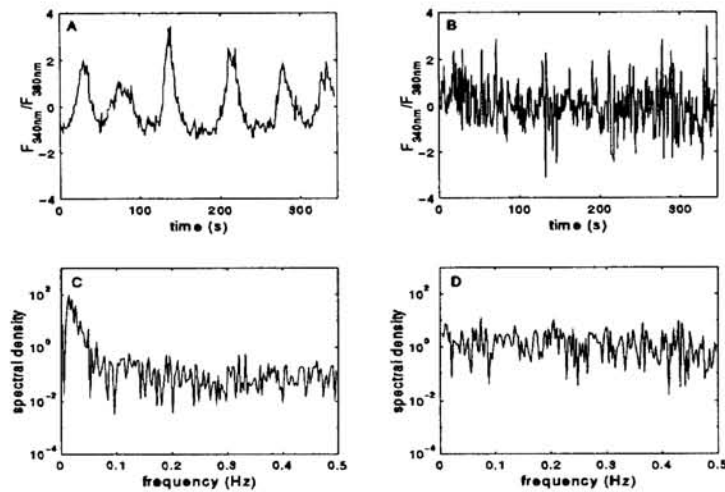

Figure 2: Results from the *independent component analysis* by the *fast fixed-point algorithm*. Two independent components of $[Ca^{2+}]_i$ were found. A: slowly oscillating $[Ca^{2+}]_i$ signal, B: fast oscillating $[Ca^{2+}]_i$ signal. Fourier power spectra of the independent components. C: the major $[Ca^{2+}]_i$ oscillatory period is 68 s, D: flat broadband power spectrum.

rylation) of target proteins (Prank *et al.*, 1998). Very recent experimental data clearly demonstrate that specificty is encoded in the frequency of $[Ca^{2+}]_i$ oscillations. Rapid oscillations of $[Ca^{2+}]_i$ are able to stimulate a set of transcription factors in T-lymphocytes whereas slow oscillations activate only one transcription factor (Dolmetsch *et al.*, 1998). Frequency-dependent gene expression is likely to be a widespread phenomenon and oscillations of $[Ca^{2+}]_i$ can occur with periods of seconds to hours. The technique of independent component analyis should be able to extract the spatio-temporal features of the $[Ca^{2+}]_i$ signal in a variety of cells and should help to understand the differential regulation of $[Ca^{2+}]_i$-dependent intracellular processes such as gene transcription or secretion.

## Acknowledgements

This study was supported by Deutsche Forschungsgemeinschaft under grants Scho 466/1-3 and Br 915/4-4.

## References

Amari, S., Cichocki, A. & Yang, H. (1996) A new learning algorithm for blind source separation. In Touretzky, D.S., Mozer, M. C. & Hasselmo, M. E., (eds.), *Advances in Neural Information Processing 8*, pp. 757-763. Cambridge, MA: MIT Press.

Bell, A. & Sejnowski, T. (1995) An information-maximization approach to blind separation and blind deconvolution. *Neural Computation* 7:1129-1159.

Berridge, M. & Galione, A. (1988) Cytosolic calcium oscillators. *FASEB* 2:3074-3082.

Cardoso, J. F. (1992) Iterative techniques for blind source separation using only fourth-order cumulants. In *Proc. EUSIPCO* (pp. 739-742). Brussels.

Comon, P. (1994) Independent component analysis - a new concept? *Signal Processing* 36:287-314.

Delfosse, N. & Loubaton, P. (1995) Adaptive blind separation of independent sources: a deflation approach. *Signal Processing* 45:59-83.

Dolmetsch, R. E., Xu, K. & Lewis, R. S. (1998) Calcium oscillations increase the efficiency and specificity of gene expression. *Nature* 392:933-936.

Hyvärinen, A. & Oja, E. (1996) A neuron that learns to separate one independent component from linear mixtures. In *Proc. IEEE Int. Conf. on Neural Networks*, pp. 62-67, Washington, D.C.

Hyvärinen, A. & Oja, E. (1997) A fast fixed-point algorithm for independent component analysis. *Neural Computation* 9:1483-1492.

Jutten, C. & Herault, J. (1991) Blind separation of sources, part I: An adaptive algorithm based on neuromimetic architecture. *Signal Processing* 24:1-10.

Moureau, E., & Macchi, O. (1993) New self-adaptive algorithms for source separation based on contrast functions. In *Proc. IEEE Signal Processing Workshop on Higher Order Statistics*, pp. 215-219, Lake Tahoe, USA.

Oja, E. & Karhunen, J. (1995) Signal separation by nonlinear hebbian learning. In Palaniswami, M., Attikiouzel, Y., Marks, R., Fogel, D. & Fukuda, T. (eds.) *Computational Intelligence - a Dynamic System Perspective* pp. 83-97. IEEE Press, New York.

Paluš, M., Schöfl, C., von zur Mühlen, A., Brabant, G. & Prank, K. (1998) Coarse-grained entropy rates quantify fast $Ca^{2+}$ dynamics modulated by pharmacological stimulation. *Pacific Symposium on Biocomputing 1998*:645-656.

Prank, K., Läer, L., Wagner, M., von zur Mühlen, A., Brabant, G. & Schöfl, C. (1998) Decoding of intracellular calcium spike trains. *Europhys. Lett.* 42:143-147.

Schöfl, C., Rössig, L., Leitolf, H., Mader, T., von zur Mühlen, A. & Brabant, G. (1996) Generation of repetitive $Ca^{2+}$ transients by bombesin requires intracellular release and influx of $Ca^{2+}$ through voltage-dependent and voltage independent channels in single HIT cells. *Cell Calcium* 19(6):485-493.

Tsien, R. W. & Tsien, R. Y. (1990) Calcium channels, stores, and oscillations. *Annu. Rev. Cell Biol.* 6:715-760.
